# From Deformations to Parts:
# Motion-based Segmentation of 3D Objects

**Soumya Ghosh**[1], **Erik B. Sudderth**[1], **Matthew Loper**[2], and **Michael J. Black**[2]

[1]Department of Computer Science, Brown University, {`sghosh,sudderth`}`@cs.brown.edu`
[2]Perceiving Systems Department, Max Planck Institute for Intelligent Systems,
{`mloper,black`}`@tuebingen.mpg.de`

## Abstract

We develop a method for discovering the parts of an articulated object from aligned meshes of the object in various three-dimensional poses. We adapt the distance dependent Chinese restaurant process (ddCRP) to allow nonparametric discovery of a potentially unbounded number of parts, while simultaneously guaranteeing a spatially connected segmentation. To allow analysis of datasets in which object instances have varying 3D shapes, we model part variability across poses via affine transformations. By placing a matrix normal-inverse-Wishart prior on these affine transformations, we develop a ddCRP Gibbs sampler which tractably marginalizes over transformation uncertainty. Analyzing a dataset of humans captured in dozens of poses, we infer parts which provide quantitatively better deformation predictions than conventional clustering methods.

## 1   Introduction

Mesh segmentation methods decompose a three-dimensional (3D) mesh, or a collection of aligned meshes, into their constituent parts. This well-studied problem has numerous applications in computational graphics and vision, including texture mapping, skeleton extraction, morphing, and mesh registration and simplification. We focus in particular on the problem of segmenting an articulated object, given aligned 3D meshes capturing various object poses. The meshes we consider are complete surfaces described by a set of triangular faces, and we seek a segmentation into spatially coherent parts whose spatial transformations capture object articulations. Applied to various poses of human bodies as in Figure 1, our approach identifies regions of the mesh that deform together, and thus provides information which could inform applications such as the design of protective clothing.

Mesh segmentation has been most widely studied as a static clustering problem, where a single mesh is segmented into "semantic" parts using low-level geometric cues such as distance and curvature [1, 2]. While supervised training data can sometimes lead to improved results [3], there are many applications where such data is unavailable, and the proper way to partition a single mesh is inherently ambiguous. By searching for parts which deform consistently across many meshes, we create a better-posed problem whose solution is directly useful for modeling objects in motion.

Several issues must be addressed to effectively segment collections of articulated meshes. First, the number of parts comprising an articulated object is unknown *a priori*, and must be inferred from the observed deformations. Second, mesh faces exhibit strong spatial correlations, and the inferred parts must be contiguous. This spatial connectivity is needed to discover parts which correspond with physical object structure, and required by target applications such as skeleton extraction. Finally, our primary goal is to understand the structure of human bodies, and humans vary widely in size and shape. People move and deform in different ways depending on age, fitness, body fat, etc. A segmentation of the human body should take into account this range of variability in the popula-

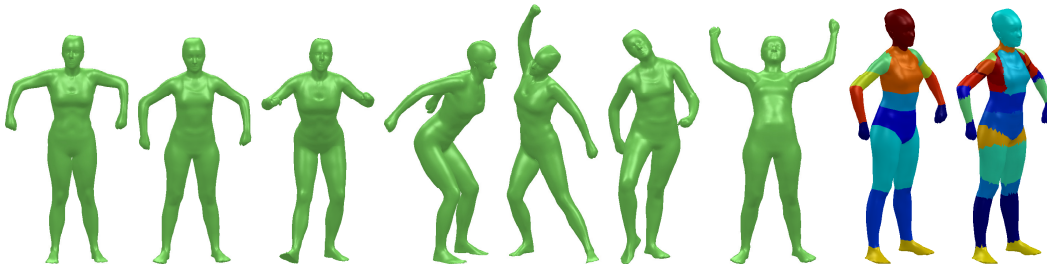

Figure 1: Human body segmentation. *Left:* Reference poses for two female bodies, and those bodies captured in five other poses. *Right:* A manual segmentation used to align these meshes [6], and the segmentation inferred by our ddCRP model from 56 poses. The ddCRP segmentation discovers parts whose motion is nearly rigid, and includes small parts such as elbows and knees absent from the manual segmentation.

tion. To our knowledge, no previous methods for segmenting meshes combine information about deformation from multiple bodies to address this *corpus segmentation* problem.

In this paper, we develop a statistical model which addresses all of these issues. We adapt the *distance dependent Chinese restaurant process* (ddCRP) [4] to model spatial dependencies among mesh triangles, and enforce spatial contiguity of the inferred parts [5]. Unlike most previous mesh segmentation methods, our Bayesian nonparametric approach allows data-driven inference of an appropriate number of parts, and uses a affine transformation-based likelihood to accommodate object instances of varying shape. After developing our model in Section 2, Section 3 develops a Gibbs sampler which efficiently marginalizes the latent affine transformations defining part deformation. We conclude in Section 4 with results examining meshes of humans and other articulated objects, where we introduce a metric for quantitative evaluation of deformation-based segmentations.

## 2 A Part-Based Model for Mesh Deformation

Consider a collection of $J$ meshes, each with $N$ triangles. For some input mesh $j$, we let $y_{jn} \in \mathbb{R}^3$ denote the 3D location of the center of triangular face $n$, and $Y_j = [y_{j1}, \ldots, y_{jN}] \in \mathbb{R}^{3 \times N}$ the full mesh configuration. Each mesh $j$ has an associated $N$-triangle reference mesh, indexed by $b_j$. We let $x_{bn} \in \mathbb{R}^4$ denote the location of triangle $n$ in reference mesh $b$, expressed in homogeneous coordinates ($x_{bn}(4) = 1$). A full reference mesh $X_b = [x_{b1}, \ldots, x_{bN}]$. In our later experiments, $Y_j$ encodes the 3D mesh for a person in pose $j$, and $X_{b_j}$ is the reference pose for the same individual.

We estimate aligned correspondences between the triangular faces of the input pose meshes $Y_j$, and the reference meshes $X_b$, using a recently developed method [6]. This approach robustly handles 3D data capturing varying shapes and poses, and outputs meshes which have equal numbers of faces in one-to-one alignment. Our segmentation model does not depend on the details of this alignment method, and could be applied to data produced by other correspondence algorithms.

### 2.1 Nonparametric Spatial Priors for Mesh Partitions

The recently proposed distance dependent Chinese restaurant process (ddCRP) [4], a generalization of the CRP underlying Dirichlet process mixture models [7], has a number of attractive properties which make it particularly well suited for modeling segmentations of articulated objects. By placing prior probability mass on partitions with arbitrary numbers of parts, it allows data-driven inference of the true number of mostly-rigid parts underlying the observed data. In addition, by choosing an appropriate distance function we can encourage spatially adjacent triangles to lie in the same part, and *guarantee* that all inferred parts are spatially contiguous [5].

The Chinese restaurant process (CRP) is a distribution on all possible partitions of a set of objects (in our case, mesh triangles). The generative process can be described via a restaurant with an infinite number of tables (in our case, parts). Customers (triangles) $i$ enter the restaurant in sequence and select a table $z_i$ to join. They pick an occupied table with probability proportional to the number of customers already sitting there, or a new table with probability proportional to a scaling parameter $\alpha$.

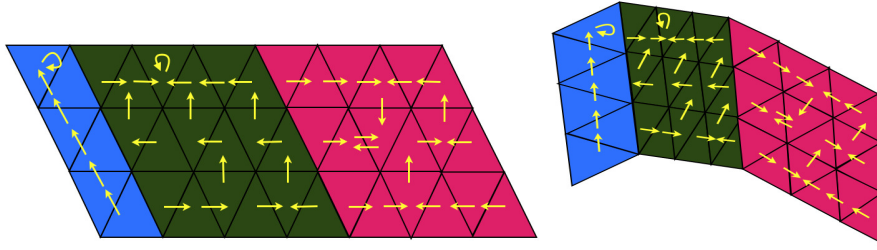

Figure 2: *Left:* A reference mesh in which links (yellow arrows) currently define three parts (connected components). *Right:* Each part undergoes a distinct affine transformation, generated as in Equation (2).

The final seating arrangement gives a partition of the data, where each occupied table corresponds to a part in the final segmentation.

Although described sequentially, the CRP induces an exchangeable distribution on partitions, for which the segmentation probability is invariant to the order in which triangle allocations are sampled. This is inappropriate for mesh data, in which nearby triangles are far more likely to lie in the same part. The ddCRP alters the CRP by modeling customer links not to tables, but to other customers. The link $c_m$ for customer $m$ is sampled according to the distribution

$$p\left(c_m = n \mid D, f, \alpha\right) \propto \begin{cases} f(d_{mn}) & m \neq n, \\ \alpha & m = n. \end{cases} \tag{1}$$

Here, $d_{mn}$ is an externally specified distance between data points $m$ and $n$, and $\alpha$ determines the probability that a customer links to themselves rather than another customer. The monotonically decreasing decay function $f(d)$ mediates how the distance between two data points affects their probability of connecting to each other. The overall link structure specifies a partition: two customers are clustered together if and only if one can reach the other by traversing the link edges.

We define the distance between two triangles as the minimal number of hops, between adjacent faces, required to reach one triangle from the other. A "window" decay function of width 1, $f(d) = \mathbf{1}[d \leq 1]$, then restricts triangles to link only to immediately adjacent faces. Note that this doesn't limit the size of parts, since all pairs of faces are potentially reachable via a sequence of adjacent links. However, it does guarantee that only spatially contiguous parts have non-zero probability under the prior. This constraint is preserved by our MCMC inference algorithm.

## 2.2 Modeling Part Deformation via Affine Transformations

Articulated object deformation is naturally described via the spatial transformations of its constituent parts. We expect the triangular faces within a part to deform according to a coherent part-specific transformation, up to independent face-specific noise. The near-rigid motions of interest are reasonably modeled as affine transformations, a family of co-linearity preserving linear transformations. We concisely denote the transformation from a reference triangle to an observed triangle via a matrix $A \in \mathbb{R}^{3 \times 4}$. The fourth column of $A$ encodes translation of the corresponding reference triangle via homogeneous coordinates $x_{bn}$, and the other entries encode rotation, scaling, and shearing.

Previous approaches have treated such transformations as parameters to be estimated during inference [8, 9]. Here, we instead define a prior distribution over affine transformations. Our construction allows transformations to be analytically marginalized when learning our part-based segmentation, but retains the flexibility to later estimate transformations if desired. Explictly modeling transformation uncertainty makes our MCMC inference more robust and rapidly mixing [7], and also allows data-driven determination of an appropriate number of parts.

The matrix of numbers encoding an affine transformation is naturally modeled via multivariate Gaussian distributions. We place a conjugate, matrix normal-inverse-Wishart [10, 11] prior on the affine transformation $A$ and residual noise covariance matrix $\Sigma$:

$$\begin{aligned} \Sigma &\sim \mathcal{IW}(n_0, S_0) \\ A \mid \Sigma &\sim \mathcal{MN}(M, \Sigma, K) \end{aligned} \tag{2}$$

Here, $n_0 \in \mathbb{R}$ and $S_0 \in \mathbb{R}^{3\times 3}$ control the variance and mean of the Wishart prior on $\Sigma^{-1}$. The mean affine transformation is $M \in \mathbb{R}^{3\times 4}$, and $K \in \mathbb{R}^{4\times 4}$ and $\Sigma$ determine the variance of the prior on $A$. Applied to mesh data, these parameters have physical interpretations and can be estimated from the data collection process. While such priors are common in Bayesian regression models, our application to the modeling of geometric affine transformations appears novel.

Allocating a different affine transformation for the motion of each part in each pose (Figure 2), the overall generative model can be summarized as follows:

1. For each triangle $n$, sample an associated link $c_n \sim \mathrm{ddCRP}\,(\alpha, f, D)$. The part assignments $z$ are a deterministic function of the sampled links $c = [c_1, \ldots, c_N]$.

2. For each pose $j$ of each part $k$, sample an affine transformation $A_{jk}$ and residual noise covariance $\Sigma_{jk}$ from the matrix normal-inverse-Wishart prior of Equation (2).

3. Given these pose-specific affine transformations and assignments of mesh faces to parts, independently sample the observed location of each pose triangle relative to its corresponding reference triangle, $y_{jn} \sim \mathcal{N}(A_{jz_n} x_{b_j n}, \Sigma_{jz_n})$.

Note that $\Sigma_{jk}$ governs the degree of non-rigid deformation of part $k$ in pose $j$. It also indirectly influences the number of inferred parts: a large $S_0$ makes large $\Sigma_{jk}$ more probable, which allows more non-rigid deformation and permits models which utilize fewer parts. The overall model is

$$p(\mathbf{Y}, c, A, \Sigma \mid \mathbf{X}, b, D, \alpha, f, \eta) = p(c \mid D, f, \alpha)$$
$$\prod_{j=1}^{J}\left[\prod_{k=1}^{K(c)} p(A_{jk}, \Sigma_{jk} \mid \eta)\right]\left[\prod_{n=1}^{N} \mathcal{N}(y_{jn} \mid A_{jz_n} x_{b_j n}, \Sigma_{jz_n})\right] \quad (3)$$

where $\mathbf{Y} = \{Y_1, \ldots, Y_J\}$, $\mathbf{X} = \{X_1, \ldots, X_B\}$, $b = [b_1, \ldots, b_J]$, the ddCRP links $c$ define assignments $z$ to $K(c)$ parts, and $\eta = \{n_0, S_0, M, K\}$ are likelihood hyperparameters. There is a single reference mesh $X_b$ for each object instance $b$, and $Y_j$ captures a single deformed pose of $X_{b_j}$.

### 2.3 Previous Work

Previous work has also sought to segment a mesh into parts based on observed articulations [8, 12, 13, 14]. The two-stage procedure of Rosman et al. [13] first minimizes a variational functional regularized to favor piecewise constant transformations, and then clusters the transformations into parts. Several other segmentation procedures [12, 14] lack coherent probabilistic models, and thus have difficulty quantifying uncertainty and determining appropriate segmentation resolutions.

Anguelov et al. [8] define a global probabilistic model, and use the EM algorithm to jointly estimate parts and their transformations. They explicitly model spatial dependencies among mesh faces, but their Markov random field cannot ensure that parts are spatially connected; a separate connected components process is required. Heuristics are used to determine an appropriate number of parts.

Ambitious recent work has considered a model for joint mesh alignment and segmentation [9]. However, this approach suffers from many of the issues noted above: the number of parts must be specified *a priori*, parts may not be contiguous, and their EM inference appears prone to local optima.

## 3 Inference

We seek the constituent parts of an articulated model, given observed data ($\mathbf{X}$, $\mathbf{Y}$, and $b$). These parts are characterized by the posterior distribution of the customer links $c$. We approximate this posterior using a collapsed Gibbs sampler, which iteratively draws $c_n$ from the conditional distribution

$$p(c_n \mid c_{-n}, \mathbf{X}, \mathbf{Y}, b, D, f, \alpha, \eta) \propto p(c_n \mid D, f, \alpha) p(\mathbf{Y} \mid z(c), \mathbf{X}, b, \eta). \quad (4)$$

Here, $z(c)$ is the clustering into parts defined by the customer links $c$. The ddCRP prior is given by Equation (1), while the likelihood term in the above equation further factorizes as

$$p(\mathbf{Y} \mid z(c), \mathbf{X}, b, \eta) = \prod_{k=1}^{K(c)} \prod_{j=1}^{J} p(Y_{jk} \mid X_{b_j k}, \eta) \quad (5)$$

where $Y_{jk} \in \mathbb{R}^{3 \times N_k}$ is the set of triangular faces in part $k$ of pose $j$, and $X_{b_jk}$ are the corresponding reference faces. Exploiting the conjugacy of the normal likelihood to the prior over affine transformations in Equation (2), we marginalize the part-specific latent variables $A_{jk}$ and $\Sigma_{jk}$ to compute the marginal likelihood in closed form (see the supplement for a derivation):

$$p(Y_{jk} \,|\, X_{b_jk}, \eta) = \frac{|K|^{3/2}|S_0|^{(n_0/2)}\Gamma_3\left(\frac{N_k+n_0}{2}\right)}{\pi^{(3N_k/2)}|S_{xx}|^{(3/2)}|S_0+S_{y|x}|^{((N_k+n_0)/2)}\Gamma_3(\frac{n_0}{2})}, \tag{6}$$

$$S_{xx} = X_{b_jk}X_{b_jk}{}^T + K, \quad S_{yx} = Y_{jk}X_{b_jk}{}^T + MK, \tag{7}$$

$$S_{y|x} = Y_{jk}Y_{jk}{}^T + MKM^T - S_{yx}(S_{xx})^{-1}S_{yx}^T. \tag{8}$$

Instead of explicitly sampling from Equation (4), a more efficient sampler [4] can be derived by observing that different realizations of the link $c_n$ only make a small change to the partition structure. First, note that removing a link $c_n$ generates a partition $z(c_{-n})$ which is either identical to the old partition $z(c)$ or contains one extra part, created by splitting some existing part. Sampling new realizations of $c_n$ will give rise to new partitions $z(c_{-n} \cup c_n^{(new)})$, which may either be identical to $z(c_{-n})$ or contain one less part, due to a merge of two existing parts. We thus sample $c_n$ from the following distribution which only tracks those parts which change with different realizations of $c_n$:

$$p(c_n \,|\, c_{-n}, \mathbf{X}, \mathbf{Y}, b, D, f, \alpha, \eta) \propto \begin{cases} p(c_n \,|\, D, f, \alpha)\Delta(\mathbf{Y}, \mathbf{X}, b, z(c), \eta) & \text{if } c_n \text{ links } k_1 \text{ and } k_2; \\ p(c_n \,|\, D, \alpha) & \text{otherwise,} \end{cases}$$

$$\Delta(\mathbf{Y}, \mathbf{X}, b, z(c), \eta) = \frac{\prod_{j=1}^{J} p(Y_{jk_1 \cup k_2} \,|\, X_{b_jk_1 \cup k_2}, \eta)}{\prod_{j=1}^{J} p(Y_{jk_1} \,|\, X_{b_jk_1}, \eta) \prod_{j=1}^{J} p(Y_{jk_2} \,|\, X_{b_jk_2}, \eta)}. \tag{9}$$

Here, $k_1$ and $k_2$ are parts in $z(c_{-n})$. Note that if the mesh segmentation $c$ is the only quantity of interest, the analytically marginalized affine transformations $A_{jk}$ need not be directly estimated. However, for some applications the transformations are of direct interest. Given a sampled segmentation, the part-specific parameters for pose $j$ have the following posterior [10]:

$$p(A_{jk}, \Sigma_{jk} \,|\, Y_j^k, X^k, \eta) \propto \mathcal{MN}(A_{jk} \,|\, S_{yx}S_{xx}^{-1}, \Sigma_{jk}, S_{xx})\mathcal{IW}(\Sigma_{jk} \,|\, N_k + n_0, S_{y|x} + S_0) \tag{10}$$

Marginalizing the noise covariance matrix, the distribution over transformations is then

$$p(A_{jk} \,|\, Y_j^k, X^k, \eta) = \int \mathcal{MN}(A_{jk} \,|\, S_{yx}S_{xx}^{-1}, \Sigma_{jk}, S_{xx})IW(\Sigma_{jk} \,|\, N_k + n_0, S_{y|x} + S_0) \, d\Sigma_{jk}$$

$$= \mathcal{MT}(A_{jk} \,|\, N_k + n_0, S_{yx}S_{xx}^{-1}, S_{xx}, S_{y|x} + S_0) \tag{11}$$

where $\mathcal{MT}(\cdot)$ is a matrix-t distribution [11] with mean $S_{yx}S_{xx}^{-1}$, and $N_k + n_0$ degrees of freedom.

# 4 Experimental Results

We now experimentally validate, both qualitatively and quantitatively, our *mesh-ddcrp* model. Because "ground truth" parts are unavailable for the real body pose datasets of primary interest, we propose an alternative evaluation metric based on the prediction of held-out object poses, and show that the mesh-ddcrp performs favorably against competing approaches.

We primarily focus on a collection of 56 training meshes, acquired and aligned [6] from 3D scans of two female subjects in 27 and 29 poses. For quantitative tests, we employ 12 meshes of each of six different female subjects [15] (Figure 4). For each subject, a mesh in a canonical pose is chosen as the reference mesh (Figure 1). These meshes contain about 20,000 faces.

## 4.1 Hyperparameter Specification and MCMC Learning

The hyperparameters that regularize our mesh-ddcrp prior have intuitive interpretations, and can be specified based on properties of the mesh data under consideration. As described in Section 2.1, the ddCRP distances $D$ and $f$ are set to guarantee spatially connected parts. The self-connection parameter is set to a small value, $\alpha = 10^{-8}$, to encourage creation of larger parts.

The matrix normal-inverse-Wishart prior on affine transformations $A_{jk}$, and residual noise covariances $\Sigma_{jk}$, has hyperparameters $\eta = \{n_0, S_0, M, K\}$. The mean affine transformation $M$ is set to

the identity transformation, because on average we expect mesh faces to undergo small deformations. For the noise covariance prior, we set the degrees of freedom $n_0 = 5$, a value which makes the prior variance nearly as large as possible while ensuring that the mean remains finite. The expected part variance $S_0$ captures the degree of non-rigidity which we expect parts to demonstrate, as well as noise from the mesh alignment process. The correspondence error in our human meshes is approximately 0.01m; allowing for some part non-rigidity, we set $\sigma = 0.015$m and $S_0 = \sigma^2 \times \mathbf{I}_{3\times3}$. $K$ is a precision matrix set to $K = \sigma^2 \times \text{diag}(1, 1, 1, 0.1)$. The Kronecker product of $K^{-1}$ and $S_0$ governs the covariance of the distribution on $A$. Our settings make this nearly identity for most components, but the translation components of $A$ have variance which is an order of magnitude larger, so that the expected scale of the translation parameters matches that of the mesh coordinates.

In our experiments, we ran the mesh-ddcrp sampler for 200 iterations from each of five random initializations, and selected the most probable posterior sample. The computational cost of a Gibbs iteration scales linearly with the number of meshes; our unoptimized Matlab implementation required around 10 hours to analyze 56 human meshes.

## 4.2 Baseline Segmentation Methods

We compare the mesh-ddcrp model to three competing methods. The first is a modified agglomerative clustering technique [16] which enforces spatial contiguity of the faces within each part. At initialization, each face is deemed to be its own part. Adjacent parts on the mesh are then merged based on the squared error in describing their motion by affine transformations. Only adjacent parts are considered in these merge steps, so that parts remain spatially connected.

Our second baseline is based on a publicly available implementation of spectral clustering methods [17], a popular approach which has been previously used for mesh segmentation [18]. We compare to an affinity matrix specifically designed to cluster faces with similar motions [19]. The affinity between two mesh faces $u$, $v$ is defined as $C_{uv} = \exp\{-\frac{\sigma_{uv}+\sqrt{m_{uv}}}{S^2}\}$, where $m_{uv} = \frac{1}{J^2}\sum_j \delta_{uvj}$, $\delta_{uvj}$ is the Euclidean distance between $u$ and $v$ in pose $j$, $\sigma_{uv} = \sqrt{\frac{1}{J}\sum_j(\delta_{uvj} - \bar{\delta}_{uv})^2}$ is the corresponding standard deviation, and $S = \frac{1}{M}\sum_{u,v}\sigma_{uv} + \sqrt{m_{uv}}$ for all $M$ pairs of faces $u, v$.

For the agglomerative and spectral clustering approaches, the number of parts must be externally specified; we experimented with $K = 5, 10, 15, 20, 25, 30$ parts. We also consider a Bayesian nonparametric baseline which replaces the ddCRP prior over mesh partitions with a standard CRP prior. The resulting *mesh-crp* model may estimate the number of parts, but doesn't model mesh structure or enforce part contiguity. The expected number of parts under the CRP prior is roughly $\alpha \log N$; we set $\alpha = 2$ so that the expected number of mesh-crp parts is similar to the number of parts discovered by the mesh-ddcrp. To exploit bilateral symmetry, for all methods we only segment the right half of each mesh. The resulting segmentation is then reflected onto the left half.

## 4.3 Part Discovery and Motion Prediction

We first consider the synthetic Tosca dataset [20], and separately analyze the Centaur (six poses) and Horse (eight poses) meshes. These meshes contain about 31,000 and 38,000 triangular faces, respectively. Figure 3 displays the segmentations of the Tosca meshes inferred by mesh-ddcrp. The inferred parts largely correspond to groups of mesh faces which undergo similar transformations.

Figure 4 displays the results produced by the ddCRP, as well as our baseline methods, on the human mesh data. Qualitatively, the segmentations produced by mesh-ddcrp correspond to our intuitions about the body. Note that in addition to capturing the head and limbs, the segmentation successfully segregates distinctly moving small regions such as knees, elbows, shoulders, biceps, and triceps. In all, the mesh-ddcrp detects 20 distinctly moving parts for one half of the body.

We now introduce a quantitative measure of segmentation quality: segmentations are evaluated by their ability to explain the articulations of test meshes with novel shapes and poses. Given a collection of $T$ test meshes $Y_t$ with corresponding reference meshes $X_{b_t}$, and a candidate segmentation into $K$ parts, we compute

$$\mathcal{E} = \frac{1}{T}\sum_{t=1}^{T}\sum_{k=1}^{K}||Y_{tk} - A_{tk}^* X_{b_t k}||_2. \tag{12}$$

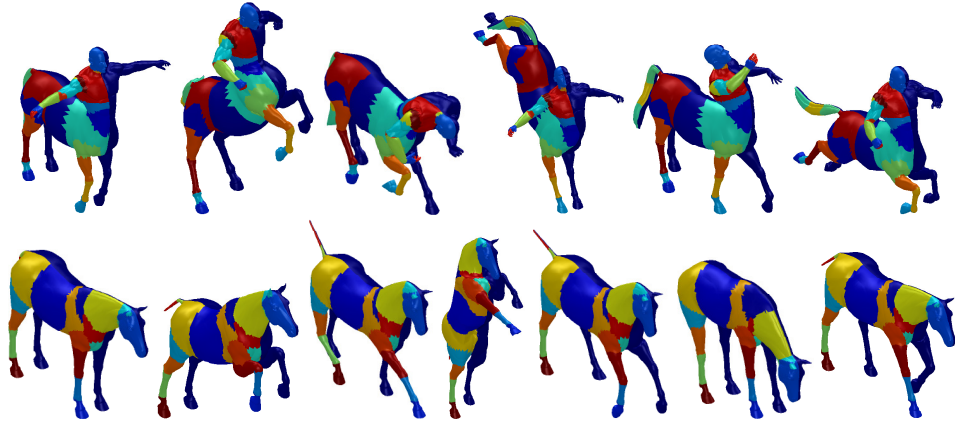

Figure 3: Segmentations produced by mesh-ddcrp on synthetic Tosca meshes [20]. The first mesh in each row displays the chosen reference mesh. For illustration, we have only segmented the right half of each mesh.

Here, $A_{tk}^*$ is the least squares estimate of the single affine transformation responsible for mapping $X_{b_t k}$ to $Y_{tk}$. Note that Equation (12) is trivially zero for a degenerate solution wherein each mesh face is assigned to its own part. However, segmentations of similar resolution may safely be compared using Equation (12), with lower errors corresponding to better segmentations.

On our test set of human meshes, the mesh-ddcrp model produces an error of $\mathcal{E} = 1.39$ meters, which corresponds to sub-millimeter accuracy when normalized by the number of faces. Figure 4 displays a plot comparing the errors achieved by the different methods. Mesh-ddcrp is significantly better than all other methods, including for settings of $K$ which allocate 50% more parts to competing approaches, according to a Wilcoxon's signed rank test (5% significance level).

Next, we demonstrate the benefits of sharing information among differently shaped bodies. We selected an illustrative articulated pose for each of the two training subjects in addition to their respective reference poses (Figure 4). The chosen poses either exhibit upper or lower body deformations, but not both. The meshes were then segmented both independently for the two subjects and jointly sharing information across subjects. Figure 5 demonstrates that the independent segmentations exhibit both undersegmented (legs in the first set) and oversegmented (head in the second) parts. However, sharing information among subjects results in parts which correspond well with physical human bodies. Note that with only two articulated poses, we are able to generate meaningful segmentations in about an hour of computation. This data-limited scenario also demonstrates the benefits of the ddCRP prior: as shown in Figure 5, the parts extracted by mesh-crp are "patchy", spatially disconnected, and physically implausible.

## 5  Discussion

Adapting the ddCRP to collections of 3D meshes, we have developed an effective approach for the discovery an unknown number of parts underlying articulated object motion. Unlike previous methods, our model guarantees that parts are spatially connected, and uses transformations to model instances with potentially varying body shapes. Via a novel application of matrix normal-inverse-Wishart priors, our sampler analytically marginalizes transformations for improved efficiency. While we have modeled part motion via affine transformations, future work should explore more accurate Lie algebra characterizations of deformation manifolds [21].

Experiments with dozens of real human body poses provide strong quantitative evidence that our approach produces state-of-the-art segmentations with many potential applications. We are currently exploring methods for using multiple samples from the ddCRP posterior to characterize part uncertainty, and scaling our Monte Carlo learning algorithms to datasets containing thousands of meshes.

**Acknowledgments**   This work was supported in part by the Office of Naval Research under contract W911QY-10-C-0172. We thank Eric Rachlin, Alex Weiss, and David Hirshberg for acquiring and aligning the human meshes, and Aggeliki Tsoli for her helpful comments.

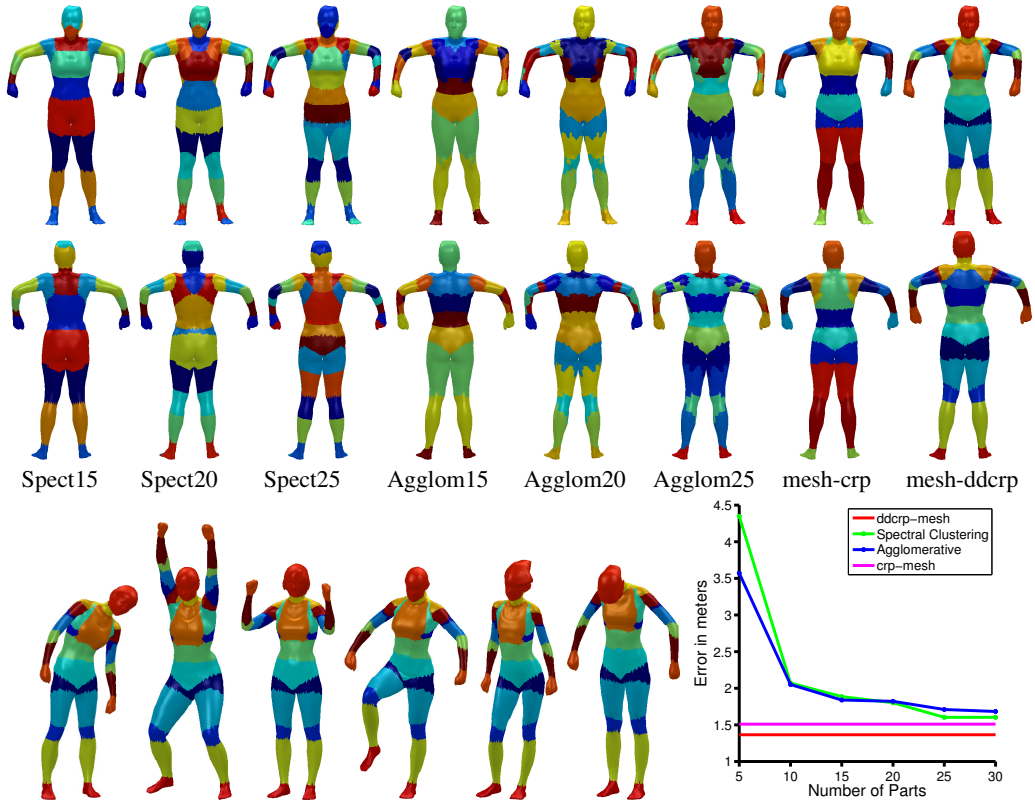

Figure 4: *Top two rows (left to right):* Segmentations produced by spectral and agglomerative clustering with 15, 20, and 25 clusters respectively, followed by the mesh-crp and mesh-ddcrp segmentations. *Bottom row:* Test set results. We display mesh-ddcrp segmentations for several test meshes, and quantitatively compare methods.

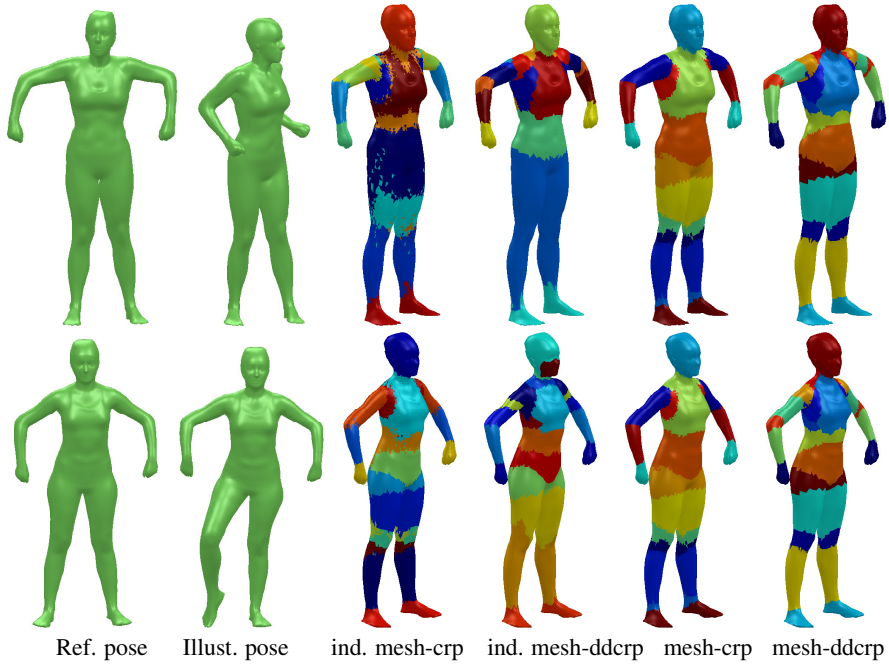

Figure 5: Impact of sharing information across bodies with varying shapes. The two rows correspond to the training subjects. Each row displays the reference pose, an illustrative articulated pose, mesh-crp and mesh-ddcrp segmentations produced by independently segmenting the pair of poses of each individual, and mesh-crp and mesh-ddcrp segmentations produced by jointly segmenting the chosen poses from both subjects.

# References

[1] M. Attene, S. Katz, M. Mortara, G. Patane, M. Spagnuolo, and A. Tal. Mesh segmentation — A comparative study. In *SMI*, 2006.

[2] Xiaobai Chen, Aleksey Golovinskiy, and Thomas Funkhouser. A benchmark for 3D mesh segmentation. *ACM Transactions on Graphics (Proc. SIGGRAPH)*, 28(3):73:1–73:12, 2009.

[3] Evangelos Kalogerakis, Aaron Hertzmann, and Karan Singh. Learning 3D Mesh Segmentation and Labeling. *ACM Transactions on Graphics*, 29(4):102:1–102:12, July 2010.

[4] David M. Blei and Peter I. Frazier. Distance dependent Chinese restaurant processes. *J. Mach. Learn. Res.*, 12:2461–2488, November 2011.

[5] S. Ghosh, A. B. Ungureanu, E. B. Sudderth, and D. Blei. Spatial distance dependent Chinese restaurant processes for image segmentation. In *NIPS*, pages 1476–1484, 2011.

[6] D. Hirshberg, M. Loper, E. Rachlin, and M.J. Black. Coregistration: Simultaneous alignment and modeling of articulated 3D shape. In *ECCV*, pages 242–255, 2012.

[7] R. M. Neal. Markov chain sampling methods for Dirichlet process mixture models. *JCGS*, 9(2):249–265, 2000.

[8] D. Anguelov, D. Koller, H. Pang, P. Srinivasan, and S. Thrun. Recovering articulated object models from 3d range data. In *UAI*, pages 18–26, 2004.

[9] J. Franco and E. Boyer. Learning temporally consistent rigidities. In *IEEE CVPR*, pages 1241–1248, 2011.

[10] E. B. Fox. *Bayesian Nonparametric Learning of Complex Dynamical Phenomena*. PhD thesis, Massachusetts Institute of Technology, Cambridge, MA, 2009.

[11] A. K. Gupta and D. K. Nagar. *Matrix Variate Distributions*. Chapman & Hall/CRC, October 2000.

[12] Tong-Yee Lee, Yu-Shuen Wang, and Tai-Guang Chen. Segmenting a deforming mesh into near-rigid components. *The Visual Computer*, 22(9):729–739, September 2006.

[13] Guy Rosman, Michael M. Bronstein, Alexander M. Bronstein, Alon Wolf, and Ron Kimmel. Group-valued regularization framework for motion segmentation of dynamic non-rigid shapes. In *SSVM'11*, pages 725–736, 2012.

[14] Stefanie Wuhrer and Alan Brunton. Segmenting animated objects into near-rigid components. *The Visual Computer*, 26:147–155, 2010.

[15] N. Hasler, C. Stoll, M. Sunkel, B. Rosenhahn, and H.-P. Seidel. A statistical model of human pose and body shape. In *Computer Graphics Forum (Proc. Eurographics 2009)*, volume 2, pages 337–346, March 2009.

[16] R. N. Shepard. Multidimensional scaling, tree-fitting, and clustering. *Science*, 210:390–398, October 1980.

[17] Wen-Yen Chen, Yangqiu Song, Hongjie Bai, Chih-Jen Lin, and Edward Y. Chang. Parallel spectral clustering in distributed systems. *IEEE PAMI*, 33(3):568–586, 2011.

[18] Rong Liu and Hao Zhang. Segmentation of 3D meshes through spectral clustering. In *Pacific Conference on Computer Graphics and Applications*, pages 298–305, 2004.

[19] Edilson de Aguiar, Christian Theobalt, Sebastian Thrun, and Hans-Peter Seidel. Automatic conversion of mesh animations into skeleton-based animations. *Computer Graphics Forum*, 27(2):389–397, 2008.

[20] Alexander Bronstein, Michael Bronstein, and Ron Kimmel. Calculus of nonrigid surfaces for geometry and texture manipulation. *IEEE Tran. on Viz. and Computer Graphics*, 13:902–913, 2007.

[21] Oren Freifeld and Michael J. Black. Lie bodies: A manifold representation of 3D human shape. In *European Conf. on Computer Vision (ECCV)*, Part I, LNCS 7572, pages 1–14. Springer-Verlag, October 2012.

